# Bayesian models of human action understanding

**Chris L. Baker, Joshua B. Tenenbaum & Rebecca R. Saxe**
{clbaker,jbt,saxe}@mit.edu
Department of Brain and Cognitive Sciences
Massachusetts Institute of Technology

## Abstract

We present a Bayesian framework for explaining how people reason about and predict the actions of an intentional agent, based on observing its behavior. Action-understanding is cast as a problem of inverting a probabilistic generative model, which assumes that agents tend to act rationally in order to achieve their goals given the constraints of their environment. Working in a simple sprite-world domain, we show how this model can be used to infer the goal of an agent and predict how the agent will act in novel situations or when environmental constraints change. The model provides a qualitative account of several kinds of inferences that preverbal infants have been shown to perform, and also fits quantitative predictions that adult observers make in a new experiment.

## 1 Introduction

A woman is walking down the street. Suddenly, she turns 180 degrees and begins running in the opposite direction. Why? Did she suddenly realize she was going the wrong way, or change her mind about where she should be headed? Did she remember something important left behind? Did she see someone she is trying to avoid? These explanations for the woman's behavior derive from taking the *intentional stance*: treating her as a rational agent whose behavior is governed by beliefs, desires or other mental states that refer to objects, events, or states of the world [5].

Both adults and infants have been shown to make robust and rapid intentional inferences about agents' behavior, even from highly impoverished stimuli. In "sprite-world" displays, simple shapes (e.g., circles) move in ways that convey a strong sense of agency to adults, and that lead to the formation of expectations consistent with goal-directed reasoning in infants [9, 8, 14]. The importance of the intentional stance in interpreting everyday situations, together with its robust engagement even in preverbal infants and with highly simplified perceptual stimuli, suggest that it is a core capacity of human cognition.

In this paper we describe a computational framework for modeling intentional reasoning in adults and infants. Interpreting an agent's behavior via the intentional stance poses a highly underconstrained inference problem: there are typically many configurations of beliefs and desires consistent with any sequence of behavior. We define a probabilistic generative model of an agent's behavior, in which behavior is dependent on hidden variables representing beliefs and desires. We then model intentional reasoning as a Bayesian inference about these hidden variables given observed behavior sequences.

It is often said that "vision is inverse graphics" – the inversion of a causal physical process of scene formation. By analogy, our analysis of intentional reasoning might be called "inverse planning", where the observer infers an agent's intentions, given observations of the agent's behavior, by inverting a model of how intentions cause behavior. The intentional stance assumes that an agent's actions depend causally on mental states via the *principle of rationality*: rational agents tend to act to achieve their desires as optimally as possible, given their beliefs. To achieve their desired goals, agents must typically not only select single actions but must construct *plans*, or sequences of intended actions. The standards of "optimal plan" may vary with agent or circumstance: possibilities include achieving goals "as quickly as possible", "as cheaply ...", "as reliably ...", and so on. We assume a soft, probabilistic version of the rationality principle, allowing that agents can often only approximate the optimal sequence of actions, and occasionally act in unexpected ways.

The paper is organized as follows. We first review several theoretical accounts of intentional reasoning from the cognitive science and artificial intelligence literatures, along with some motivating empirical findings. We then present our computational framework, grounding the discussion in a specific sprite-world domain. Lastly, we present results of our model on two sprite-world examples inspired by previous experiments in developmental psychology, and results of the model on our own experiments.

## 2  Empirical studies of intentional reasoning in infants and adults

### 2.1  Inferring an invariant goal

The ability to predict how an agent's behavior will adapt when environmental circumstances change, such as when an obstacle is inserted or removed, is a critical aspect of intentional reasoning. Gergely, Csibra and colleagues [8, 4] showed that preverbal infants can infer an agent's goal that appears to be invariant across different circumstances, and can predict the agent's future behavior by effectively assuming that it will act to achieve its goal in an efficient way, subject to the constraints of its environment. Their experiments used a looking-time (violation-of-expectation) paradigm with sprite-world stimuli. Infant participants were assigned to one of two groups. In the "obstacle" condition, infants were habituated to a sprite (a colored circle) moving ("jumping") in a curved path over an obstacle to reach another object. The size of the obstacle varied across trials, but the sprite always followed a near-shortest path over the obstacle to reach the other object. In the "no obstacle" group, infants were habituated to the sprite following the same curved "jumping" trajectory to the other object, but without an obstacle blocking its path. Both groups were then presented with the same test conditions, in which the obstacle was placed out of the sprite's way, and the sprite followed either the old, curved path or a new direct path to the other object. Infants from the "obstacle" group looked longer at the sprite following the unobstructed curved path, which (in the test condition) was now far from the most efficient route to the other object. Infants in the "no obstacle" group looked equally at both test stimuli. That is, infants in the "obstacle" condition appeared to interpret the sprite as moving in a rational goal-directed fashion, with the other object as its goal. They expected the sprite to plan a path to the goal that was maximally efficient, subject to environmental constraints when present. Infants in the "no obstacle" group appeared more uncertain about whether the sprite's movement was actually goal-directed or about what its goal was: was it simply to reach the other object, or something more complex, such as reaching the object via a particular curved path?

### 2.2  Inferring goals of varying complexity: rational means-ends analysis

Gergely et al. [6], expanding on work by Meltzoff [11], showed that infants can infer goals of varying complexity, again by interpreting agents' behaviors as rational responses to environmental constraints. In two conditions, infants saw an adult demonstrate an unfamiliar complex action: illuminating a light-box by pressing its top with her forehead. In the "hands occupied" condition, the demonstrator pretended to be cold and wrapped a blanket

around herself, so that she was incapable of using a more typical means (i.e., her hands) to achieve the same goal. In the "hands free" condition the demonstrator had no such constraint. Most infants in the "hands free" condition spontaneously performed the head-press action when shown the light-box one week later, but only a few infants in the "hands occupied" condition did so; the others illuminated the light-box simply by pressing it with their hands. Thus infants appear to assume that rational agents will take the most efficient path to their goal, and that if an agent appears to systematically employ an inefficient means, it is likely because the agent has adopted a more complex goal that includes not only the end state but also the means by which that end should be achieved.

## 2.3 Inductive inference in intentional reasoning

Gergely and colleagues interpret their findings as if infants are reasoning about intentional action in an almost logical fashion, deducing the goal of an agent from its observed behavior, the rationality principle, and other implicit premises. However, from a computational point of view, it is surely oversimplified to think that the intentional stance could be implemented in a deductive system. There are too many sources of uncertainty and the inference problem is far too underconstrained for a logical approach to be successful. In contrast, our model posits that intentional reasoning is probabilistic. People's inferences about an agent's goal should be graded, reflecting a tradeoff between the prior probability of a candidate goal and its likelihood in light of the agent's observed behavior. Inferences should become more confident as more of the agent's behavior is observed.

To test whether human intentional reasoning is consistent with a probabilistic account, it is necessary to collect data in greater quantities and with greater precision than infant studies allow. Hence we designed our own sprite-world experimental paradigm, to collect richer quantitative judgments from adult observers. Many experiments are possible in this paradigm, but here we describe just one study of statistical effects on goal inference.

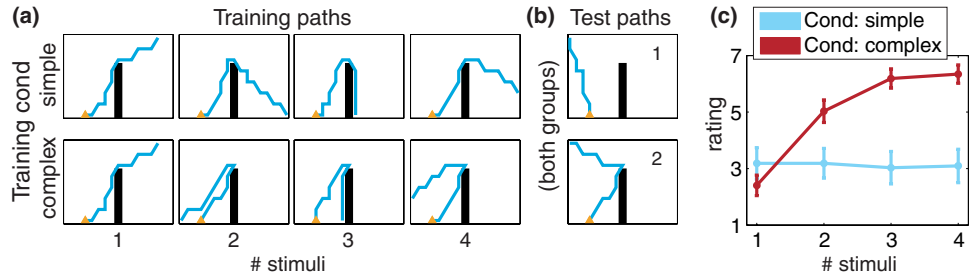

Figure 1: (a) Training stimuli in complex and simple goal conditions. (b) Test stimuli 1 and 2. Test stimuli was the same for each group. (c) Mean of subjects' ratings with standard error bars (n=16).

Sixteen observers were told that they would be watching a series of animations of a mouse running in a simple maze (a box with a single internal wall). The displays were shown from an overhead perspective, with an animated schematic trace of the mouse's path as it ran through the box. In each display, the mouse was placed in a different starting location and ran to recover a piece of cheese at a fixed, previously learned location. Observers were told that the mouse had learned to follow a more-or-less direct path to the cheese, regardless of its starting location. Subjects saw two conditions in counterbalanced order. In one condition ("simple goal"), observers saw four displays consistent with this prior knowledge. In another condition ("complex goal"), observers saw movements suggestive of a more complex, path-dependent goal for the mouse: it first ran directly to a particular location in the middle of the box (the "via-point"), and only then ran to the cheese. Fig. 1(a) shows the mouse's four trajectories in each of these conditions. Note that the first trajectory was the same in both conditions, while the next three were different. Also, all four trajectories in both conditions passed through the same hypothetical via-point in the middle of the box, which was not marked in any conspicuous way. Hence both the simple goal ("get to

the cheese") and complex goal ("get to the cheese via point $X$") were logically possible interpretations in both conditions.

Observers' interpretations were assessed after viewing each of the four trajectories, by showing them diagrams of two test paths (Fig. 1(b)) running from a novel starting location to the cheese. They were asked to rate the probability of the mouse taking one or the other test path using a 1-7 scale: 1 = definitely path 1, 7 = definitely path 2, with intermediate values expressing intermediate degrees of confidence. Observers in the simple-goal condition always leaned towards path 1, the direct route that was consistent with the given prior knowledge. Observers in the complex-goal condition initially leaned just as much towards path 1, but after seeing additional trajectories they became increasingly confident that the mouse would follow path 2 (Fig. 1(c)). Importantly, the latter group increased its average confidence in path 2 with each subsequent trajectory viewed, consistent with the notion that goal inference results from something like a Bayesian integration process: prior probability favors the simple goal, but successive observations are more likely under the complex goal.

## 3   Previous models of intentional reasoning

The above phenomena highlight two capacities than any model of intentional reasoning should capture. First, representations of agents' mental states should include at least primitive planning capacities, with a constrained space of candidate goals and subgoals (or intended paths) that can refer to objects or locations in space, and the tendency to choose action sequences that achieve goals as efficiently as possible. Second, inferences about agents' goals should be probabilistic, and be sensitive to both prior knowledge about likely goals as well as statistical evidence for more complex or less likely goals that better account for observed actions.

These two components are clearly not sufficient for a complete account of human intentional reasoning, but most previous accounts do not include even these capacities. Gergely, Csibra and colleagues [7] have proposed an informal (noncomputational) model in which agents are essentially treated as rational planners, but inferences about agents' goals are purely deductive, without a role for probabilistic expectations or gradations of confidence.

A more statistically sophisticated computational framework for inferring goals from behavior has been proposed by [13], but this approach does not incorporate planning capacities. In this framework, the observer learns to represent an agent's policies, conditional on the agent's goals. Within a static environment, this knowledge allows an observer to infer the goal of an agent's actions, predict subsequent actions, and perform imitation, but it does not support generalization to new environments where the agent's policy must adapt in response. Further, because generalization is not based on strong prior knowledge such as the principle of rationality, many observations are needed for good performance. Likewise, probabilistic approaches to plan recognition in AI (e.g., [3, 10]) typically represent plans in terms of policies (state-action pairs) that do not generalize when the structure of the environment changes in some unexpected way, and that require much data to learn from observations of behavior.

Perhaps closest to how people reason with the intentional stance are methods for *inverse reinforcement learning* (IRL) [12], or methods for learning an agent's utility function [2]. Both approaches assume a rational agent who maximizes expected utility, and attempt to infer the agent's utility function from observations of its behavior. However, the utility functions that people attribute to intentional agents are typically much more structured and constrained than in conventional IRL. Goals are typically defined as relations towards objects or other agents, and may include subgoals, preferred paths, or other elements. In the next section we describe a Bayesian framework for modeling intentional reasoning that is similar in spirit to IRL, but more focused on the kinds of goal structures that are cognitively natural to human adults and infants.

## 4 The Bayesian framework

We propose to model intentional reasoning by combining the inferential power of statistical approaches to action understanding [12, 2, 13] with simple versions of the representational structures that psychologists and philosophers [5, 7] have argued are essential in theory of mind. This section first presents our general approach, and then presents a specific mathematical model for the "mouse" sprite-world introduced above.

Most generally, we assume a world that can be represented in terms of entities, attributes, and relations. Some attributes and relations are *dynamic*, indexed by a time dimension. Some entities are *agents*, who can perform actions at any time $t$ with the potential to change the world state at time $t+1$. We distinguish between environmental state, denoted $W$, and agent states, denoted $S$. For simplicity, we will assume that there is exactly one intentional agent in the world, and that the agent's actions can only affect its own state $s \in S$. Let $s_{0:T}$ be a sequence of $T+1$ agent states. Typically, observations of multiple state sequences of the agent are available, and in general each may occur in a separate environment. Let $s_{0:T}^{1:N}$ be a set of $N$ state sequences, and let $w^{1:N}$ be a set of $N$ corresponding environments. Let $A_s$ be the set of actions available to the agent from state $s$, and let $C(a)$ be the cost to the agent of action $a \in A_s$. Let $P(s_{t+1}|a_t, s_t, w)$ be the distribution over the agent's next state $s_{t+1}$, given the current state $s_t$, an action $a_t \in A_{s_t}$, and the environmental state $w$.

The agent's actions are assumed to depend on mental states such as *beliefs* and *desires*. In our context, beliefs correspond to knowledge about the environmental state. Desires may be simple or complex. A simple desire is an *end goal*: a world state or class of states that the agent will act to bring about. There are many possibilities for more complex goals, such as achieving a certain end by means of a certain route, achieving a certain sequence of states in some order, and so on. We specify a particular goal space $G$ of simple and complex goals for sprite-worlds in the next subsection. The agent draws goals $g \in G$ from a prior distribution $P(g|w^{1:N})$, which constrains goals to be feasible in the environments $w^{1:N}$ from which observations of the agent's behavior are available.

Given the agent's goal $g$ and an environment $w$, we can define a value $V_{g,w}(s)$ for each state $s$. The value function can be defined in various ways depending on the domain, task, and agent type. We specify a particular value function in the next subsection that reflects the goal structure of our sprite-world agent. The agent is assumed to choose actions according to a probabilistic policy, with a preference for actions with greater expected increases in value. Let $Q_{g,w}(s,a) = \sum_{s'} P(s'|a,s,w)V_{g,w}(s') - C(a)$ be the expected value of the state resulting from action $a$, minus the cost of the action. The agent's policy is

$$P(a_t|s_t, g, w) \propto \exp(\beta Q_{g,w}(s_t, a_t)). \tag{1}$$

The parameter $\beta$ controls how likely the agent is to select the most valuable action. This policy embodies a "soft" principle of rationality, which allows for inevitable sources of suboptimal planning, or unexplained deviations from the direct path. A graphical model illustrating the relationship between the environmental state, and the agent's goals, actions, and states is shown in Fig. 2.

The observer's task is to infer $g$ from the agent's behavior. We assume that state sequences are independent given the environment and the goal. The observer infers $g$ from $s_{0:T}^{1:N}$ via Bayes' rule, conditional on $w^{1:N}$:

$$P(g|s_{0:T}^{1:N}, w^{1:N}) \propto P(g|w^{1:N}) \prod_{i=1}^{N} P(s_{0:T}^i|g, w^i). \tag{2}$$

We assume that state transition probabilities and action probabilities are conditionally independent given the agent's goal $g$, the agent's current state $s_t$, and the environment $w$. The likelihood of a state sequence $s_{0:T}$ given a goal $g$ and an environment $w$ is computed by marginalizing over possible actions generating state transitions:

$$P(s_{0:T}|g, w) = \prod_{t=0}^{T-1} \sum_{a_t \in A_{s_t}} P(s_{t+1}|a_t, s_t, w)P(a_t|s_t, g, w). \tag{3}$$

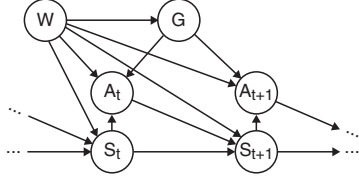

Figure 2: Two time-slice dynamic Bayes net representation of our model, where $W$ is the environmental state, $G$ is the agent's goal, $S_t$ is the agent's state at time $t$, and $A_t$ is the agent's action at time $t$. Beliefs, desires, and actions intuitively map onto $W$, $G$ and $A$, respectively.

## 4.1 Modeling sprite-world inferences

Several additional assumptions are necessary to apply the above framework to any specific domain, such as the sprite-worlds discussed in §2. The size of the grid, the location of obstacles, and likely goal points (such as the location of the cheese in our experimental stimuli) are represented by $W$, and assumed to be known to both the agent and the observer. The agent's state space $S$ consists of valid locations in the grid. All state sequences are assumed to be of the same length. The action space $A_s$ consists of moves in all compass directions $\{N, S, E, W, NE, NW, SE, SW\}$, except where blocked by an obstacle, and action costs are Euclidean. The agent can also choose to remain still with cost 1. We assume $P(s_{t+1}|a_t, s_t, w)$ takes the agent to the desired adjacent grid point deterministically.

The set of possible goals $G$ includes both simple and complex goals. Simple goals will just be specific end states in $S$. While many kinds of complex goals are possible, we assume here that a complex goal is just the combination of a desired end state with a desired means to achieving that end. In our sprite-worlds, we identify "desired means" with a constraint that the agent must pass through an additional specified location enroute, such as the via-point in the experiment from §2.3. Because the number of complex goals defined in this way is much larger than the number of simple goals, the likelihood of each complex goal is small relative to the likelihood of individual simple goals. In addition, although path-dependent goals are possible, they should not be likely a priori. We thus set the prior $P(g|w^{1:N})$ to favor simple goals by a factor of $\gamma$. For simplicity, we assume that the agent draws just a single invariant goal $g \in G$ from $P(g|w^{1:N})$, and we assume that this prior distribution is known to the observer. More generally, an agent's goals may vary across different environments, and the prior $P(g|w^{1:N})$ may have to be learned.

We define the value of a state $V_{g,w}(s)$ as the expected total cost to the agent of achieving $g$ while following the policy given in Eq. 1. We assume the desired end-state is absorbing and cost-free, which implies that the agent attempts the *stochastic shortest path* (with respect to its probabilistic policy) [1]. If $g$ is a complex goal, $V_{g,w}(s)$ is based on the stochastic shortest path through the specified via-point. The agent's value function is computed using the value iteration algorithm [1] with respect to the policy given in Eq. 1.

Finally, to compare our model's predictions with behavioral data from human observers, we must specify how to compute the probability of novel trajectories $s'_{0:T}$ in a new environment $w'$, such as the test stimuli in Fig. 1, conditioned on an observed sequence $s_{0:T}$ in environment $w$. This is just an average over the predictions for each possible goal $g$:

$$P(s'_{0:T}|s_{0:T}, w, w') = \sum_{g \in G} P(s'_{0:T}|g, w')P(g|s_{0:T}, w, w'). \qquad (4)$$

# 5 Sprite-world simulations

## 5.1 Inferring an invariant goal

As a starting point for testing our model, we return to the experiments of Gergely et al. [8, 4, 7], reviewed in §2.1. Our input to the model, shown in Fig. 3(a,b), differs slightly from the original stimuli used in [8], but the relevant details of interest are spared: goal-directed action in the presence of constraints. Our model predictions, shown in Fig. 3(c), capture the qualitative results of these experiments, showing a large contrast between the straight path and the curved path in the condition with an obstacle, and a relatively small contrast in the condition with no obstacle. In the "no obstacle" condition, our model infers that the agent has a more complex goal, constrained by a via-point. This significantly increases the

probability of the curved test path, to the point where the difference between the probability of observing curved and straight paths is negligible.

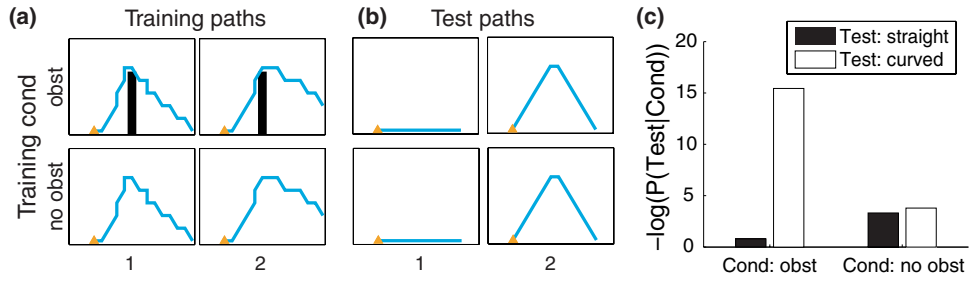

Figure 3: Inferring an invariant goal. (a) Training input in obstacle and no obstacle conditions. (b) Test input is the same in each condition. (c) Model predictions: negative log likelihoods of test paths 1 and 2 given data from training condition. In the obstacle condition, a large dissociation is seen between path 1 and path 2, with path 1 being much more likely. In the no obstacle condition, there is not a large preference for either path 1 or path 2, qualitatively matching Gergely et al.'s results [8].

## 5.2 Inferring goals of varying complexity: rational means-ends analysis

Our next example is inspired by the studies of Gergely et al. [6] described in §2.2. In our sprite-world version of the experiment, we varied the amount of evidence for a simple versus a complex goal, by inputting the same three trajectories with and without an obstacle present (Fig. 4(a)). In the "obstacle" condition, the trajectories were all approximately shortest paths to the goal, because the agent was forced to take indirect paths around the obstacle. In the "no obstacle" condition, no such constraint was present to explain the curved paths. Thus a more complex goal is inferred, with a path constrained to pass through a via-point. Given a choice of test paths, shown in Fig. 4(b), the model shows a double-dissociation between the probability of the direct path and the curved path through the putative via-point, given each training condition (Fig. 4(c)), similar to the results in [6].

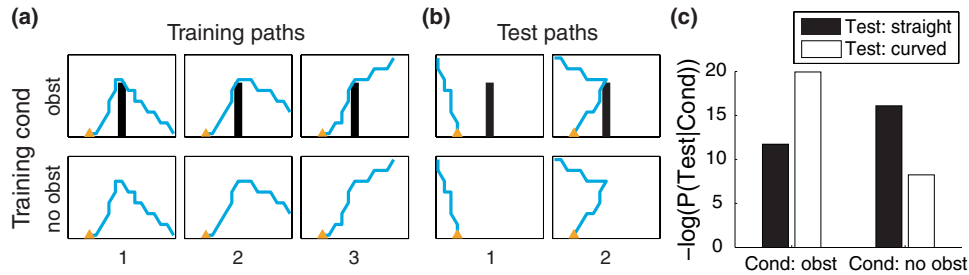

Figure 4: Inferring goals of varying complexity. (a) Training input in obstacle and no obstacle conditions. (b) Test input in each condition. (c) Model predictions: a double dissociation between probability of test paths 1 and 2 in the two conditions. This reflects a preference for the straight path in the first condition, where there is an obstacle to explain the agent's deflections in the training input, and a preference for the curved path in the second condition, where a complex goal is inferred.

## 5.3 Inductive inference in intentional reasoning

Lastly, we present the results of our model on our own behavioral experiment, first described in §2.3 and shown in Fig. 1. These data demonstrated the statistical nature of people's intentional inferences. Fig. 5 compares people's judgments of the probability that the agent takes a particular test path with our model's predictions. To place model predictions and human judgments on a comparable scale, we fit a sigmoidal psychometric transformation to the computed log posterior odds for the curved test path versus the straight path. The Bayesian model captures the graded shift in people's expectations in the "complex goal" condition, as evidence accumulates that the agent always seeks to pass through an arbitrary via-point enroute to the end state.

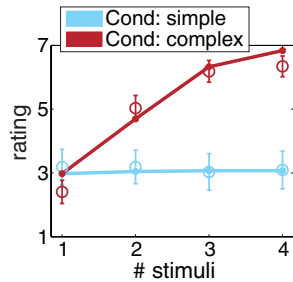

Figure 5: Experimental results: model fit for behavioral data. Mean ratings are plotted as hollow circles. Error bars give standard error. The log posterior odds from the model were fit to subjects' ratings using a scaled sigmoid function with range $(1, 7)$. The sigmoid function includes bias and gain parameters, which were fit to the human data by minimizing the sum-squared error between the model predictions and mean subject ratings.

## 6 Conclusion

We presented a Bayesian framework to explain several core aspects of intentional reasoning: inferring the goal of an agent based on observations of its behavior, and predicting how the agent will act when constraints or initial conditions for action change. Our model captured basic qualitative inferences that even preverbal infants have been shown to perform, as well as more subtle quantitative inferences that adult observers made in a novel experiment. Two future challenges for our computational framework are: representing and learning multiple agent types (e.g. rational, irrational, random, etc.), and representing and learning hierarchically structured goal spaces that vary across environments, situations and even domains. These extensions will allow us to further test the power of our computational framework, and will support its application to the wide range of intentional inferences that people constantly make in their everyday lives.

**Acknowledgments:** We thank Whitman Richards, Konrad Körding, Kobi Gal, Vikash Mansinghka, Charles Kemp, and Pat Shafto for helpful comments and discussions.

## References

[1] D. P. Bertsekas. *Dynamic Programming and Optimal Control*. Athena Scientific, Belmont, MA, 2nd edition, 2001.

[2] U. Chajewska, D. Koller, and D. Ormoneit. Learning an agent's utility function by observing behavior. In *Proc. of the 18th Intl. Conf. on Machine Learning (ICML)*, pages 35–42, 2001.

[3] E. Charniak and R. Goldman. A probabilistic model of plan recognition. In *Proc. AAAI*, 1991.

[4] G. Csibra, G. Gergely, S. Biró, O. Koós, and M. Brockbank. Goal attribution without agency cues: the perception of 'pure reason' in infancy. *Cognition*, 72:237–267, 1999.

[5] D. C. Dennett. *The Intentional Stance*. Cambridge, MA: MIT Press, 1987.

[6] G. Gergely, H. Bekkering, and I. Király. Rational imitation in preverbal infants. *Nature*, 415:755, 2002.

[7] G. Gergely and G. Csibra. Teleological reasoning in infancy: the naïve theory of rational action. *Trends in Cognitive Sciences*, 7(7):287–292, 2003.

[8] G. Gergely, Z. Nádasdy, G. Csibra, and S. Biró. Taking the intentional stance at 12 months of age. *Cognition*, 56:165–193, 1995.

[9] F. Heider and M. A. Simmel. An experimental study of apparent behavior. *American Journal of Psychology*, 57:243–249, 1944.

[10] L. Liao, D. Fox, and H. Kautz. Learning and inferring transportation routines. In *Proc. AAAI*, pages 348–353, 2004.

[11] A. N. Meltzoff. Infant imitation after a 1-week delay: Long-term memory for novel acts and multiple stimuli. *Developmental Psychology*, 24:470–476, 1988.

[12] A. Y. Ng and S. Russell. Algorithms for inverse reinforcement learning. In *Proc. of the 17th Intl. Conf. on Machine Learning (ICML)*, pages 663–670, 2000.

[13] R. P. N. Rao, A. P. Shon, and A. N. Meltzoff. A Bayesian model of imitation in infants and robots. In *Imitation and Social Learning in Robots, Humans, and Animals*. (in press).

[14] B. J. Scholl and P. D. Tremoulet. Perceptual causality and animacy. *Trends in Cognitive Sciences*, 4(8):299–309, 2000.